# English Alphabet Recognition with Telephone Speech

**Mark Fanty, Ronald A. Cole and Krist Roginski**
Center for Spoken Language Understanding
Oregon Graduate Institute of Science and Technology
19600 N.W. Von Neumann Dr., Beaverton, OR 97006

## Abstract

A recognition system is reported which recognizes names spelled over the telephone with brief pauses between letters. The system uses separate neural networks to locate segment boundaries and classify letters. The letter scores are then used to search a database of names to find the best scoring name. The speaker-independent classification rate for spoken letters is 89%. The system retrieves the correct name, spelled with pauses between letters, 91% of the time from a database of 50,000 names.

## 1 INTRODUCTION

The English alphabet is difficult to recognize automatically because many letters sound alike; e.g., B/D, P/T, V/Z and F/S. When spoken over the telephone, the information needed to discriminate among several of these pairs, such as F/S, P/T, B/D and V/Z, is further reduced due to the limited bandwidth of the channel

Speaker-independent recognition of spelled names over the telephone is difficult due to variability caused by channel distortions, different handsets, and a variety of background noises. Finally, when dealing with a large population of speakers, dialect and foreign accents alter letter pronunciations. An R from a Boston speaker may not contain an [r].

Human classification performance on telephone speech underscores the difficulty of the problem. We presented each of ten listeners with 3,197 spoken letters in random order for identification. The letters were taken from 100 telephone calls

in which the English alphabet was recited with pauses between letters, and 100 different telephone calls with first or last names spelled with pauses between letters. Our subjects averaged 93% correct classification of the letters, with performance ranging from 90% to 95%. This compares to error rates of about 1% for high quality microphone speech [DALY 87].

Over the past three years, our group at OGI has produced a series of letter classification and name retrieval systems. These systems combine speech knowledge and neural network classification to achieve accurate spoken letter recognition [COLE 90, FANTY 91]. Our initial work focused on speaker-independent recognition of isolated letters using high quality microphone speech. By accurately locating segment boundaries and carefully designing feature measurements to discriminate among letters, we achieved 96% classification of letters.

We extended isolated letter recognition to recognition of words spelled with brief pauses between the letters, again using high quality speech [FANTY 91, COLE 91]. This task is more difficult than recognition of isolated letters because there are "pauses" within letters, such as the closures in "X," "H" and "W," which must be distinguished from the pauses that separate letters, and because speakers do not always pause between letters when asked to do so. In the system, a neural network segments speech into a sequence of broad phonetic categories. Rules are applied to the segmentation to locate letter boundaries, and the hypothesized letters are re-classified using a second neural network. The letter scores from this network are used to retrieve the best scoring name from a database of 50,000 last names. First choice name retrieval was 95.3%, with 99% of the spelled names in the top three choices. Letter recognition accuracy was 90%.

During the past year, with support from US WEST Advanced Technologies, we have extended our approach to recognition of names spelled over the telephone. This report describes the recognition system, some experiments that motivated its design, and its current performance.

## 1.1 SYSTEM OVERVIEW

**Data Capture and Signal Processing.** Telephone speech is sampled at 8 kHz at 14-bit resolution. Signal processing routines perform a seventh order PLP (Perceptual Linear Predictive) analysis [HERMANSKY 90] every 3 msec using a 10 msec window. This analysis yields eight coefficients per frame, including energy.

**Phonetic Classification.** Frame-based phonetic classification provides a sequence of phonetic labels that can be used to locate and classify letters. Classification is performed by a fully-connected three-layer feed-forward network that assigns 22 phonetic category scores to each 3 msec time frame. The 22 labels provide an intermediate level of description, in which some phonetic categories, such as [b]-[d], [p]-[t]-[k] and [m]-[n] are combined; these fine phonetic distinctions are performed during letter classification, described below. The input to the network consists of 120 features representing PLP coefficients in a 432 msec window centered on the frame to be classified.

The frame-by-frame outputs of the phonetic classifier are converted to a sequence of phonetic segments corresponding to a sequence of hypothesized letters. This is

done with a Viterbi search that uses duration and phoneme sequence constraints provided by letter models. For example, the letter model for MN consists of optional glottalization (MN-q), followed by the vowel [eh] (MN-eh), followed by the nasal murmur (MN-mn). Because background noise is often classified as [f]-[s] or [m]-[n], a noise "letter" model was added which consists of either of these phonemes.

**Letter Classification.**    Once letter segmentation is performed, a set of 178 features is computed for each letter and used by a fully-connected feed-forward network with one hidden layer to reclassify the letter. Feature measurements are based on the phonetic boundaries provided by the segmentation. At present, the features consist of segment durations, PLP coefficients for thirds of the consonant (fricative or stop) before the first sonorant, PLP for sevenths of the first sonorant, PLP for the 200 msecs after the sonorant, PLP slices 6 and 10 msec after the sonorant onset, PLP slices 6 and 30 msec before any internal sonorant boundary (e.g. [eh]/[m]), zero crossing and amplitude profiles from 180 msec before the sonorant to 180 msec after the sonorant. The outputs of the classifier are the 26 letters plus the category "not a letter."

**Name Retrieval.**    The output of the classifier is a score between 0 and 1 for each letter. These scores are treated as probabilities and the most likely name is retrieved from the database of 50,000 last names. The database is stored in an efficient tree structure. Letter deletions and insertions are allowed with a penalty.

# 2   SYSTEM DEVELOPMENT

## 2.1   DATA COLLECTION

Callers were solicited through local newspaper and television coverage, and notices on computer bulletin boards and news groups. Callers had the choice of using a local phone number or toll-free 800-number.

A Gradient Technology Desklab attached to a UNIX workstation was programmed to answer the phone and record the answers to pre-recorded questions. The first three thousand callers were given the following instructions, designed to generate spoken and spelled names, city names, and yes/no responses: (1) What city are you calling from? (2) What is your last name? (3) Please spell your last name. (4) Please spell your last name with short pauses between letters. (5) Does your last name contain the letter "A" as in apple? (6) What is your first name? (7) Please spell your first name with short pauses between letters. (8) What city and state did you grow up in? (9) Would you like to receive more information about the results of this project?

In order to achieve sufficient coverage of rare letters, the final 1000 speakers were asked to recite the entire English alphabet with brief pauses between letters.

The system described here was trained on 800 speakers and tested on 400 speakers. The training set contains 400 English alphabets and 800 first and last names spelled with pauses between letters. The test set consists of 100 alphabets and 300 last names spelled with pauses between letters.

A subset of the data was phonetically labeled to train and evaluate the neural network segmenter. Time-aligned phonetic labels were assigned to 300 first and last names and 100 alphabets, using the following labels: cl bcl dcl kcl pcl tcl q aa ax ay b ch d ah eh ey f iy jh k l m n ow p r s t uw v w y z h#. This label set represents a subset of the TIMIT [LAMEL 86] labels sufficient to describe the English alphabet.

## 2.2   FRAME-BASED CLASSIFICATION

Explicit location of segment boundaries is an important feature of our approach. Consider, for example, the letters B and D. They are distinguished by information at the onset of the letter; the spectrum of the release burst of [b] and [d], and the formant transitions during the first 10 or 15 msec of the vowel [iy]. By precisely locating the burst onset and vowel onset, feature measurements can be designed to optimize discrimination. Moreover, the duration of the initial consonant segment can be used to discriminate B from P, and D from T.

A large number of experiments were performed to improve segmentation accuracy. [ROGINSKI 91]. These experiments focused on (a) determining the appropriate set of phonetic categories, (b) determining the set of features that yield the most accurate classification of these categories, and (c) determining the best strategy for sampling speech frames within the phonetic categories.

**Phonetic Categories.**  Given our recognition strategy of first locating segment boundaries and then classifying letters, it makes little sense to attempt to discriminate [b]-[d], [p]-[t]-[k] or [m]-[n] at this stage. Experiments confirmed that using the complete set of phonetic categories found in the English alphabet did not produce the most accurate frame-based phonetic classification. The actual choice of categories was guided initially by perceptual confusions in the listening experiment, and was refined through a series of experiments in which different combinations of acoustically similar categories were merged.

**Features Used for Classification.**  A series of experiments was performed which covaried the amount of acoustic context provided to the network and the number of hidden units in the network. The results are shown in Figure 1. A network with 432 msec of spectral information, centered on the frame to be classified, and 40 hidden units was chosen as the best compromise.

**Sampling of Speech Frames.**  The training and test sets contained about 1.7 million 3 msec frames of speech; too many to train on all of them The manner in which speech frames were sampled was found to have a large effect of performance. It was necessary to sample more speech frames from less frequently occurring categories and those with short durations (e.g., [b]).

The *location* within segments of the speech frames selected was found to have a profound effect on the accuracy of boundary location. Accurate boundary placement required the correct proportion of speech frames sampled near segment boundaries. For example, in order to achieve accurate location of stop bursts, it was necessary to sample a high proportion of speech frames just prior to the burst (within the

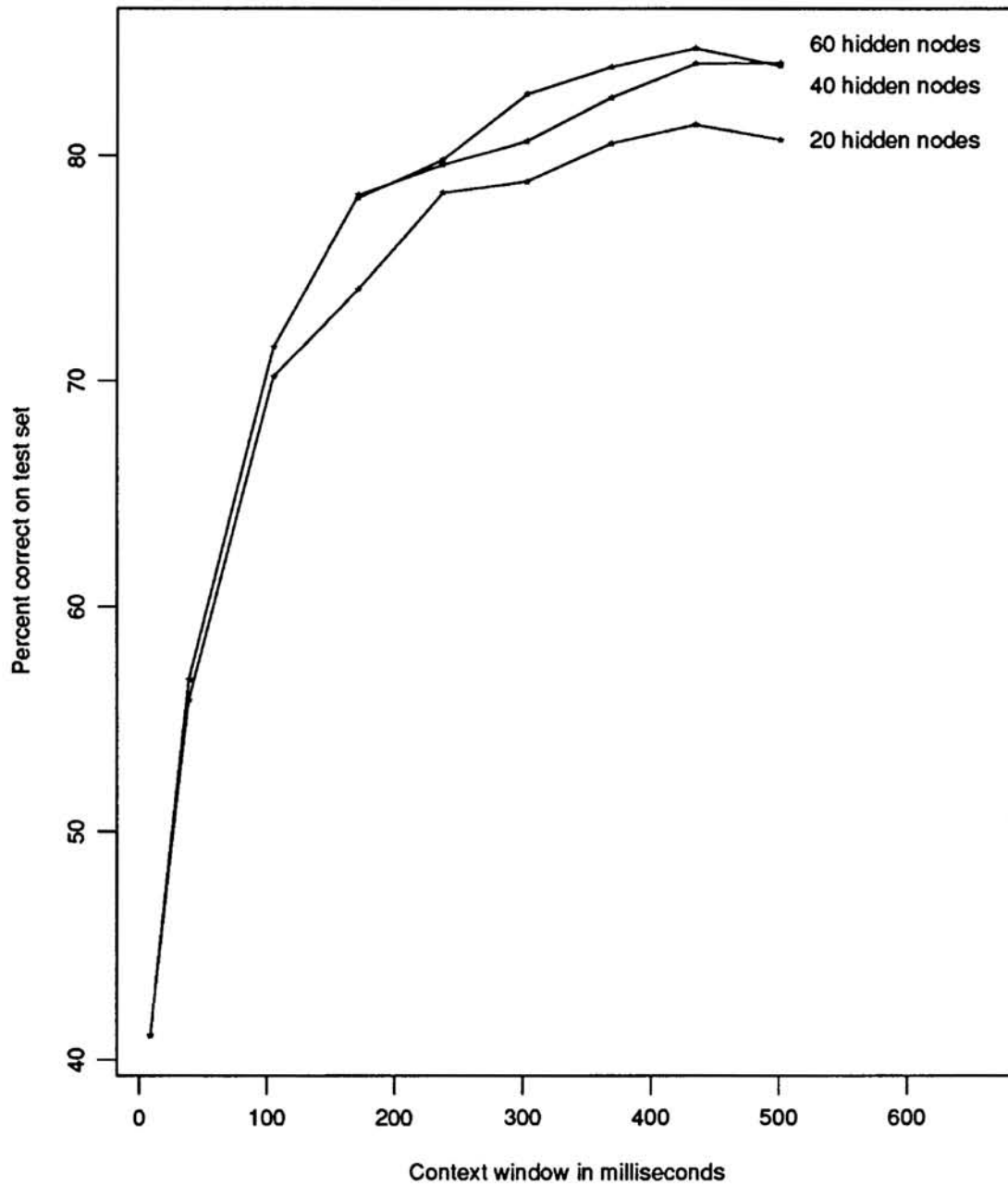

Figure 1: Performance of the phonetic classifier as a function of PLP context and number of hidden units.

closure category). Figure 2 shows the improvement in the placement of the [b]/[iy] boundary after sampling more training frames near that boundary.

## 2.3   LETTER CLASSIFICATION

In order to avoid segmenting training data for letter classification by hand, an automatic procedure was used. Each utterance was listened to and the letter names were transcribed manually. Segmentation was performed as described above, except the Viterbi search was forced to match the transcribed letter sequence. This resulted in very accurate segmentation.

One concern with this procedure was that artificially good segmentation for the training data could hurt performance on the test set, where there are bound to be more segmentation errors (since the letter sequence is not known). The letter classifier should be able to recover from segmentation errors (e.g. a B being segmented as V with a long [v] before the burst). To do so, the network must be trained with errorful segmentation.

The solution is to perform two segmentations. The forced segmentation finds the letter boundaries so the correct identity is known. A second, unforced, segmentation is performed and these phonetic boundaries are used to generate features used to train the classifier.

Any "letters" found by the unforced search which correspond to noise or silence from the forced search are used as training data for the "not a letter" category. So there are two ways noise can be eliminated: It can match the noise model of the segmenter during the Viterbi search, or it can match a letter during segmentation, but be reclassified as "not a letter" by the letter classifier. Both are necessary in the current system.

## 3   PERFORMANCE

**Frame-Based Phonetic Classification.**   The phonetic classifier was trained on selected speech frames from 200 speakers. About 450 speech frames were selected from 50 different occurrences of each phonetic category. Phonetic segmentation performance on 50 alphabets and 150 last names was evaluated by comparing the first-choice of the classifier at each time frame to the label provided by a human expert. The frame-by-frame agreement was 80% before the Viterbi search and 90% after the Viterbi search.

**Letter Classification and Name Retrieval.**   The training set consists of 400 alphabets spelled by 400 callers plus first and last names spelled by 400 callers, all with pauses between the letters.

When tested on 100 alphabets from new speakers, the letter classification was 89% with less than 1% insertions. When tested on 300 last names from new speakers, the letter classification was 87% with 1.5% insertions.

For the 300 callers spelling their last name, 90.7% of the names were correctly retrieved from a list of 50,000 common last names. 95.7% of the names were in the

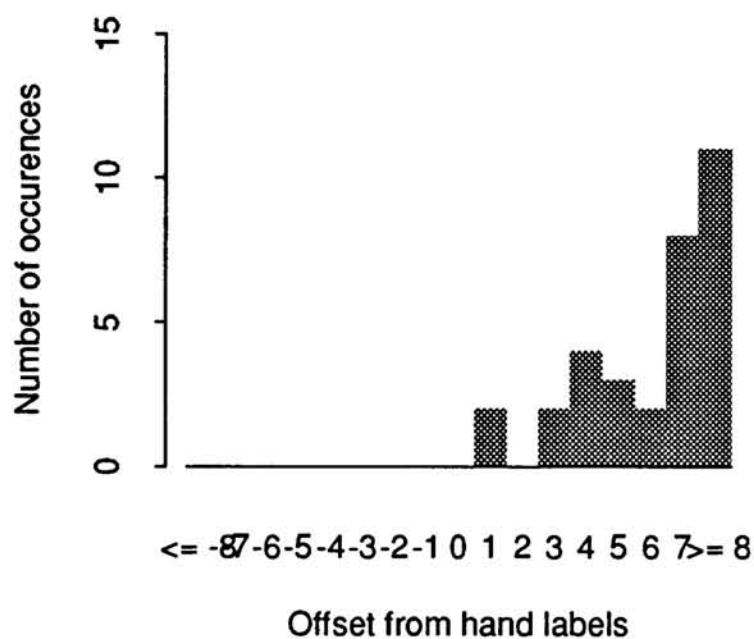

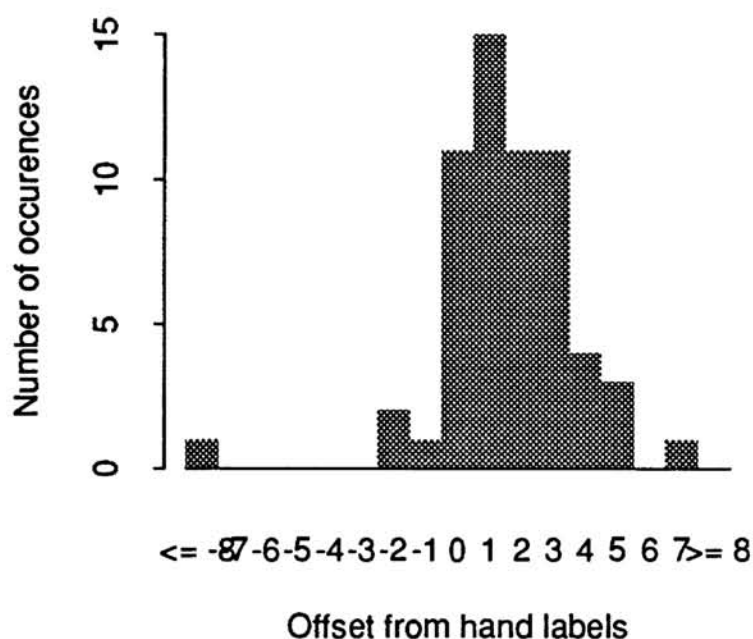

Figure 2: Test set improvement in the placement of the [b]/[iy] boundary after sampling more training frames near that boundary. The top histogram shows the difference between hand-labeled boundaries and the system's boundaries in 3 msec frames before adding extra boundary frames. The bottom histogram shows the difference after adding the boundary frames.

top three.

## 4   DISCUSSION

The recognition system described in this paper classifies letters of the English alphabet produced by any speaker over telephone lines at 89% accuracy for spelled alphabets and retrieves names from a list of 50,000 with 91% first choice accuracy.

The system has a number of characteristic features. We represent speech using an auditory model—Perceptual Linear Predictive (PLP) analysis. We perform explicit segmentation of the speech signal into phonetic categories. Explicit segmentation allows us to use segment durations to discriminate letters, and to extract features from specific regions of the signal. Finally, speech knowledge is used to design a set of features that work best for English letters. We are currently analyzing errors made by our system. The great advantage of our approach is that individual errors can be analyzed, and individual features can be added to improve performance.

### Acknowledgements

Research supported by US WEST Advanced Technologies, APPLE Computer Inc., NSF, ONR, Digital Equipment Corporation and Oregon Advanced Computing Institute.

## References

[COLE 91] R. A. Cole, M. Fanty, M. Gopalakrishnan, and R. D. T. Janssen. Speaker-independent name retrieval from spellings using a database of 50,000 names. In *Proceedings of the IEEE International Conference on Acoustics, Speech, and Signal Processing*, 1991.

[COLE 90] R. A. Cole, M. Fanty, Y. Muthusamy, and M. Gopalakrishnan. Speaker-independent recognition of spoken English letters. In *Proceedings of the International Joint Conference on Neural Networks*, San Diego, CA, 1990.

[DALY 87] N. Daly. Recognition of words from their spellings: Integration of multiple knowledge sources. Master's thesis, Massachusetts Institute of Technology, May, 1987.

[FANTY 91] M. Fanty and R. A. Cole. Spoken letter recognition. In R. P. Lippman, J. Moody, and D. S. Touretzky, editors, *Advances in Neural Information Processing Systems 3*. San Mateo, CA: Morgan Kaufmann, 1991.

[HERMANSKY 90] H. Hermansky. Perceptual Linear Predictive (PLP) analysis of speech. *J. Acoust. Soc. Am.*, 87(4):1738-1752, 1990.

[LAMEL 86] L. Lamel, R. Kassel, and S. Seneff. Speech database development: Design and analysis of the acoustic-phonetic corpus. In *Proceedings of the DARPA Speech Recognition Workshop*, pages 100-110, 1986.

[ROGINSKI 91] Krist Roginski. A neural network phonetic classifier for telephone spoken letter recognition. Master's thesis, Oregon Graduate Institute, 1991.
